# Adaptive Discriminative Generative Model and Its Applications

**Ruei-Sung Lin**[†]    **David Ross**[‡]    **Jongwoo Lim**[†]    **Ming-Hsuan Yang**[*]
[†]University of Illinois    [‡]University of Toronto    [*]Honda Research Institute
rlin1@uiuc.edu dross@cs.toronto.edu jlim1@uiuc.edu myang@honda-ri.com

## Abstract

This paper presents an adaptive discriminative generative model that generalizes the conventional Fisher Linear Discriminant algorithm and renders a proper probabilistic interpretation. Within the context of object tracking, we aim to find a discriminative generative model that best separates the target from the background. We present a computationally efficient algorithm to constantly update this discriminative model as time progresses. While most tracking algorithms operate on the premise that the object appearance or ambient lighting condition does not significantly change as time progresses, our method adapts a discriminative generative model to reflect appearance variation of the target and background, thereby facilitating the tracking task in ever-changing environments. Numerous experiments show that our method is able to learn a discriminative generative model for tracking target objects undergoing large pose and lighting changes.

## 1   Introduction

Tracking moving objects is an important and essential component of visual perception, and has been an active research topic in computer vision community for decades. Object tracking can be formulated as a continuous state estimation problem where the unobservable states encode the locations or motion parameters of the target objects, and the task is to infer the unobservable states from the observed images over time. At each time step, a tracker first predicts a few possible locations (i.e., hypotheses) of the target in the next frame based on its prior and current knowledge. The prior knowledge includes its previous observations and estimated state transitions. Among these possible locations, the tracker then determines the most likely location of the target object based on the new observation. An attractive and effective prediction mechanism is based on Monte Carlo sampling in which the state dynamics (i.e., transition) can be learned with a Kalman filter or simply modeled as a Gaussian distribution. Such a formulation indicates that the performance of a tracker is largely based on a good observation model for validating all hypotheses. Indeed, learning a robust observation model has been the focus of most recent object tracking research within this framework, and is also the focus of this paper.

Most of the existing approaches utilize static observation models and construct them before a tracking task starts. To account for all possible variation in a static observation model, it is imperative to collect a large set of training examples with the hope that it covers all possible variations of the object's appearance. However, it is well known that the appearance of an object varies significantly under different illumination, viewing angle, and shape deformation. It is a daunting, if not impossible, task to collect a training set that enumerates all possible cases. An alternative approach is to develop an adaptive method that contains a number of trackers that track different features or parts of a target object [3]. Therefore,

even though each tracker may fail under certain circumstances, it is unlikely all of them fail at the same time. The tracking method then adaptively selects the trackers that are robust at current situation to predict object locations. Although this approach improves the flexibility and robustness of a tracking method, each tracker has a static observation model which has to be trained beforehand and consequently restricts its application domains severely. There are numerous cases, e.g., robotics applications, where the tracker is expected to track a previously unseen target once it is detected. To the best of our knowledge, considerably less attention is paid to developing adaptive observation models to account for appearance variation of a target object (e.g., pose, deformation) or environmental changes (e.g., lighting conditions and viewing angles) as tracking task progresses.

Our approach is to learn a model for determining the probability of a predicted image location being generated from the class of the target or the background. That is, we formulate a binary classification problem and develop a discriminative model to distinguish observations from the target class and the background class. While conventional discriminative classifiers simply predict the class of each test sample, a good model within the above-mentioned tracking framework needs to select the most likely sample that belongs to target object class from a set of samples (or hypotheses). In other words, an observation model needs a classifier with proper probabilistic interpretation.

In this paper, we present an adaptive discriminative generative model and apply it to object tracking. The proposed model aims to best separate the target and the background in the ever-changing environment. The problem is formulated as a density estimation problem, where the goal is, given a set of positive (i.e., belonging to the target object class) and negative examples (i.e., belonging to the background class), to learn a distribution that assigns high probability to the positive examples and low probability to the negative examples. This is done in a two-stage process. First, in the generative stage, we use a probabilistic principal component analysis to model the density of the positive examples. The result of this state is a Gaussian, which assigns high probability to examples lying in the linear subspace which captures the most variance of the positive examples. Second, in the discriminative stage, we use negative examples (specifically, negative examples that are assigned high probability by our generative model) to produce a new distribution which reduces the probability of the negative examples. This is done by learning a linear projection that, when applied to the data and the generative model, increases the distance between the negative examples and the mean. Toward that end, it is formulated as an optimization problem and we show that this is a direct generalization of the conventional Fisher Linear Discriminant algorithm with proper probabilistic interpretation. Our experimental results show that our algorithm can reliably track moving objects whose appearance changes under different poses, illumination, and self deformation.

## 2 Probabilistic Tracking Algorithm

We formulate the object tracking problem as a state estimation problem in a way similar to [5] [9]. Denote $o_t$ as an image region observed at time $t$ and $O_t = \{o_1, \ldots, o_t\}$ is a set of image regions observed from the beginning to time $t$. An object tracking problem is a process to infer state $s_t$ from observation $O_t$, where state $s_t$ contains a set of parameters referring to the tracked object's 2-D position, orientation, and scale in image $o_t$. Assuming a Markovian state transition, this inference problem is formulated with a recursive equation:

$$p(s_t|O_t) = kp(o_t|s_t) \int p(s_t|s_{t-1})p(s_{t-1}|O_{t-1})ds_{t-1} \tag{1}$$

where $k$ is a constant, and $p(o_t|s_t)$ and $p(s_t|s_{t-1})$ correspond to the observation model and dynamic model, respectively.

In (1), $p(s_{t-1}|O_{t-1})$ is the state estimation given all the prior observations up to time $t-1$, and $p(o_t|s_t)$ is the likelihood that observing image $o_t$ at state $s_t$. Put together, the posterior estimation $p(s_t|O_t)$ can be computed efficiently. For object tracking, an ideal distribution

of $p(s_t|O_t)$ should peak at $o_t$, i.e., $s_t$ matching the observed object's location $o_t$. While the integral in (1) predicts the regions where object is likely to appear given all the prior observations, the observation model $p(o_t|s_t)$ determines the most likely state that matches the observation at time $t$.

In our formulation, $p(o_t|s_t)$ measures the probability of observing $o_t$ as a sample being generated by the target object class. Note that $O_t$ is an image sequence and if the images are acquired at high frame rate, it is expected that the difference between $o_t$ and $o_{t-1}$ is small though object's appearance might vary according to different of viewing angles, illuminations, and possible self-deformation. Instead of adopting a complex static model to learn $p(o_t|s_t)$ for all possible $o_t$, a simpler model suffices by adapting this model to account for the appearance changes. In addition, since $o_t$ and $o_{t-1}$ are most likely similar and computing $p(o_t|s_t)$ depends on $p(o_{t-1}|s_{t-1})$, the prior information $p(o_{t-1}|s_{t-1})$ can be used to enhance the distinctiveness between the object and its background in $p(o_t|s_t)$.

The idea of using an adaptive observation model for object tracking and then applying discriminative analysis to better predict object location is the focus of the rest the paper. The observation model we use is based on probabilistic principle component analysis (PPCA) [10]. Object Tracking using PCA models have been well exploited in the computer vision community [2]. Nevertheless, most existing tracking methods do not update the observation models as time progresses. In this paper, we follow the work by Tipping and Bishop [10] and propose an adaptive observation model based on PCA within a formal probabilistic framework. Our result is a generalization of the conventional Fisher Linear Discriminant with proper probabilistic interpretation.

## 3 A Discriminative Generative Observation Model

In this work, we track a target object based on its observations in the videos, i.e., $o_t$. Since the size of image region $o_t$ might change according to different $s_t$, we first convert $o_t$ to a standard size and use it for tracking. In the following, we denote $y_t$ as the standardized appearance vector of $o_t$.

The dimensionality of the appearance vector $y_t$ is usually high. In our experiments, the standard image size is a $19 \times 19$ patch and thus $y_t$ is a 361-dimensional vector. We thus model the appearance vector with a graphical model of low-dimensional latent variables.

### 3.1 A Generative Model with Latent Variables

A latent model relates a $n$-dimensional appearance vector $y$ to a $m$-dimensional vector of latent variables $x$:

$$y = Wx + \mu + \epsilon \tag{2}$$

where $W$ is a $n \times m$ projection matrix associating $y$ and $x$, $\mu$ is the mean of $y$, and $\epsilon$ is additive noise. As commonly assumed in factor analysis [1] and other graphical models [6], the latent variables $x$ are independent with unit variance, $x \sim \mathcal{N}(0, I_m)$, where $I_m$ is the $m$-dimensional identity matrix, and $\epsilon$ is zero mean Gaussian noise, $\epsilon \sim \mathcal{N}(0, \sigma^2 I_n)$. Since $x$ and $\epsilon$ are both Gaussians, it follows that $y$ is also a Gaussian distribution, $y \sim \mathcal{N}(\mu, C)$, where $C = WW^T + \sigma^2 I$ and $I_n$ is an $n$-dimensional identity matrix. Together with (2), we have a generative observation model:

$$p(o_t|s_t) = p(y_t|W, \mu, \epsilon) \sim \mathcal{N}(y_t|\mu, WW^T + \sigma^2 I_n) \tag{3}$$

This latent variable model follows the form of probabilistic principle component analysis, and its parameters can be estimated from a set of examples [10]. Given a set of appearance samples $Y = \{y_1, \ldots, y_N\}$, the covariance matrix of $Y$ is denoted as $S = \frac{1}{N} \sum (y - \mu)(y - \mu)^T$. Let $\{\lambda_i|i = 1, \ldots, N\}$ be the eigenvalues of $S$ arranged in descending order, i.e., $\lambda_i \geq \lambda_j$ if $i < j$. Also, define the diagonal matrix $\Sigma_m = \text{diag}(\lambda_1, \ldots, \lambda_m)$, and let $U_m$ be the eigenvectors that corresponds to the eigenvalues in $\Sigma_m$. Tipping and Bishop

[10] show that the the maximum likelihood estimate of $\mu$, $W$ and $\epsilon$ can be obtained by

$$\mu = \frac{1}{N}\sum_{i=1}^{N}y_i, \quad W = U_m(\Sigma_m - \sigma^2 I_m)^{1/2}R, \quad \sigma^2 = \frac{1}{n-m}\sum_{i=m+1}^{n}\lambda_i \qquad (4)$$

where $R$ is an arbitrary $m \times m$ orthogonal rotation matrix.

To model all possible appearance variations of a target object (due to pose, illumination and view angle change), one could resort to a mixture of PPCA models. However, it not only requires significant computation for estimating the model parameters but also leads to other serious questions such as the number of components as well as under-fitting or over-fitting. On the other hand, at any given time a linear PPCA model suffices to model gradual appearance variation if the model is constantly updated. In this paper, we use a single PPCA, and dynamically adapt the model parameters $W$, $\mu$, and $\sigma^2$ to account for appearance change.

### 3.1.1 Probability computation with Probabilistic PCA

Once the model parameters are known, we can compute the probability that a vector $y$ is a sample of this generative appearance model. From (4), the log-probability is computed by

$$\mathcal{L}(W, \mu, \sigma^2) = -\frac{1}{2}\left(N\log 2\pi + \log|C| + \overline{y}^T C^{-1}\overline{y}\right) \qquad (5)$$

where $\overline{y} = y - \mu$. Neglecting the constant terms, the log-probability is determined by $\overline{y}^T C^{-1}\overline{y}$. Together with $C = WW^T + \sigma^2 I_n$ and (4), it follows that

$$\overline{y}^T C^{-1}\overline{y} = \overline{y}^T U_m \Sigma_m^{-1} U_m^T \overline{y} + \frac{1}{\sigma^2}\overline{y}^T(I_n - U_m U_m^T)\overline{y} \qquad (6)$$

Here $\overline{y}^T U_m \Sigma_m^{-1} U_m^T \overline{y}$ is the Mahalanobis distance of $y$ in the subspace spanned by $U_m$, and $\overline{y}^T(I_n - U_m U_m^T)\overline{y}$ is the shortest distance from $y$ to this subspace spanned by $U_m$. Usually $\sigma$ is set to a small value, and consequently the probability will be determined solely by the distance to the subspace. However, the choice of $\sigma$ is not trivial. From (6), if the $\sigma$ is set to a value much smaller than the actual one, the distance to the subspace will be favored and ignore the contribution of Mahalanobis distance, thereby rendering an inaccurate estimate. The choice of $\sigma$ is even more critical in situations where the appearance changes dynamically and requires $\sigma$ to be adjusted accordingly. This topic will be further examined in the following section.

### 3.1.2 Online Learning of Probabilistic PCA

Unlike the analysis in the previous section where model parameters are estimated based on a fixed set of training examples, our generative model has to learn and update its parameters on line. Starting with a single example (the appearance of the tracked object in the first video frame), our generative model constantly updates its parameters as new observations arrive.

The equations for updating parameters are derived from (4). The update procedure of $U_m$ and $\Sigma_m$ is complicated since it involves the computations of eigenvalues and eigenvectors. Here we use a forgetting factor $\gamma$ to put more weights on the most recent data. Denote the newly arrived samples at time $t$ as $Y = \{y^1, \ldots, y^M\}$, and assume the mean $\mu$ is fixed, $U_m^t$ and $\Sigma_m^t$ can be obtained by performing singular value decomposition (SVD) on

$$[\sqrt{\gamma}U_{m,t-1}(\Sigma_{m,t-1})^{1/2}|\sqrt{(1-\gamma)}\overline{Y}] \qquad (7)$$

where $\overline{Y} = [y^1 - \mu, \ldots, y^M - \mu]$. $\Sigma_{m,t}^{1/2}$ and $U_{m,t}$ will contain the $m$-largest singular values and the corresponding singular vectors respectively at time $t$. This update procedure can be efficiently implemented using the R-SVD algorithm, e.g., [4] [7].

If the mean $\mu$ constantly changes, the above update procedure can not be applied. We recently proposed a method [8] to compute SVD with correct updated mean in which $\Sigma_{m,t}^{1/2}$

and $U_{m,t}$ can be obtained by computing SVD on

$$\left[\sqrt{\gamma}U_{m,t-1}(\Sigma_{m,t-1})^{1/2} \middle| \sqrt{(1-\gamma)}\overline{Y} \middle| \sqrt{(1-\gamma)\gamma}(\mu_{t-1}-\mu_Y)\right] \quad (8)$$

where $\overline{Y} = [y^1 - \mu_Y, \ldots, y^M - \mu_Y]$ and $\mu_Y = \frac{1}{M}\sum_{i=1}^{M} y^i$. This formulation is similar to the SVD computation with the fixed mean case, and the same incremental SVD algorithm can be used to compute $\Sigma_{m,t}^{1/2}$ and $U_{m,t}$ with an extra term shown in (8).

Computing and updating $\sigma$ is more difficult than the form in (8). In the previous section, we show that an inaccurate value of $\sigma$ will severely affect probability estimates. In order to have an accurate estimate of $\sigma$ using (4), a large set of training examples is usually required. Our generative model starts with a single example and gradually adapts the model parameters. If we update $\sigma$ based on (4), we will start with a very small value of $\sigma$ since there are only a few samples at our disposal at the outset, and the algorithm could quickly lose track of the target because of an inaccurate probability estimate. Since the training examples are not permanently stored in memory, $\lambda_i$ in (4) and consequently $\sigma$ may not be accurately updated if the number of drawn samples is insufficient. These constraints lead us to develop a method that adaptively adjusts $\sigma$ according to the newly arrived samples, which will be explained in the next section.

### 3.2 Discriminative Generative Model

As is observed in Section 2, the object's appearance at $o_{t-1}$ and $o_t$ do not change much. Therefore, we can use the observation at $o_{t-1}$ to boost the likelihood measurement in $o_t$. That is, we draw a set samples (i.e., image patches) parameterized by $\{s_{t-1}^i | i = 1, ..., k\}$ in $o_{t-1}$ that have large $p(o_{t-1}|s_{t-1}^i)$, but the low posterior $p(s_{t-1}^i|O_{t-1})$. These are treated as the negative samples (i.e., samples that are not generated from the class of the target object) that the generative model is likely to confuse at $O_t$.

Given a set samples $Y' = \{y^1, \ldots, y^k\}$ where $y^i$ is the appearance vector collected in $o_{t-1}$ based on state parameter $s_{t-1}^i$, we want to find a linear projection $V^*$ that projects $Y'$ onto a subspace such that the likelihood of $Y'$ in the subspace is minimized. Let $V$ be a $p \times n$ matrix and since $p(y|W,\mu,\sigma)$ is a Gaussian, $p(Vy|V,W,\mu,\sigma) \sim \mathcal{N}(V\mu, VCV^T)$ is a also Gaussian. The log likelihood is computed by

$$\mathcal{L}(V, W, \mu, \sigma) = -\frac{k}{2}\left(p\log(2\pi) + \log|VCV^T| + tr((VCV^T)^{-1}VS'V^T)\right) \quad (9)$$

where $S' = \frac{1}{k}\sum_{i=1}^{k}(y^i - \mu)(y^i - \mu)^T$.

To facilitate the following analysis we first assume $V$ projects $Y'$ to a 1-D space, i.e., $p = 1$ and $V = v^T$, and thus

$$\mathcal{L}(V, W, \mu, \sigma) = -\frac{k}{2}\left(\log(2\pi) + \log|v^T Cv| + \frac{v^T S'v}{v^T Cv}\right) \quad (10)$$

Note that $v^T Cv$ is the variance of the object samples in the projected space, and we need to impose a constraint, e.g., $v^t Cv = 1$, to ensure that the minimum likelihood solution of $v$ does not increase the variance in the projected space. Let $v^T Cv = 1$, the optimization problem becomes

$$v^* = \arg\max_{\{v|v^T Cv=1\}} v^T S'v = \arg\max_v \frac{v^T S'v}{v^T Cv} \quad (11)$$

Thus, we obtain an equation exactly like the Fisher discriminant analysis for a binary classification problem. In (11), $v$ is a projection that keeps the object's samples in the projected space close to the $\mu$ (with variance $v^T Cv = 1$), while keeping negative samples in $Y'$ away from $\mu$. The optimal value of $v$ is the generalized eigenvector of $S'$ and $C$ that corresponds to largest eigenvalue. In a general case, it follows that

$$V^* = \arg\max_{\{VCV^T=I\}} |VS'V^T| = \arg\max_V \frac{|VS'V^T|}{|VCV^T|} \quad (12)$$

where $V^*$ can be obtained by solving a generalized eigenvalue problem of $S'$ and $C$. By projecting observation samples onto a low dimensional subspace, we enhance the discriminative power of the generative model. In the meanwhile, we reduce the time required to compute probabilities, which is also a critical improvement for real time applications like object tracking.

### 3.2.1 Online Update of Discriminative Analysis

The computation of the projection matrix $V$ depends on matrices $C$ and $S'$. In section 3.1.2, we have shown the procedures to update $C$. The same procedures can be used to update $S'$. Let $\mu_{Y'} = \frac{1}{k}\sum_{i=1}^{k} y^i$ and $S_{Y'} = \frac{1}{k}\sum_{i=1}^{k}(y^i - \mu_{Y'})(y^i - \mu_{Y'})^T$,

$$S' = \frac{1}{k}\sum_{i=1}^{k}(y^i - \mu)(y^i - \mu)^T = S_y + (\mu - \mu_{Y'})(\mu - \mu_{Y'})^T \tag{13}$$

Given $S'$ and $C$, $V$ is computed by solving a generalized eigenvalue problem. If we decompose $S' = A^T A$ and $C = B^T B$, then we can find $V$ more efficiently using generalized singular value decomposition. Denote $U_{Y'}$ and $\Sigma_{Y'}$ as the SVD of $S_{Y'}$, it follows that by letting $A = [U_{Y'}\Sigma_{Y'}^{1/2} \mid (\mu - \mu_{Y'})]^T$ and $B = [U_m\Sigma_m^{1/2} \mid \sigma^2 I]^T$, we obtain $S' = A^T A$ and $C = B^T B$.

As is detailed in [4], $V$ can be computed by first performing a QR factorization:

$$\begin{bmatrix} A \\ B \end{bmatrix} = \begin{bmatrix} Q_A \\ Q_B \end{bmatrix} R \tag{14}$$

and computing the singular value decomposition of $Q_A$

$$Q_A = U_A D_A V_A^T \tag{15}$$

, we then obtain $V = R^{-1}V_A$. The rank of $A$ is usually small in vision applications, and $V$ can be computed efficiently, thereby facilitating tracking the process.

## 4 Proposed Tracking Algorithm

In this section, we summarize the proposed tracking algorithm and demonstrate how the abovementioned learning and inference algorithms are incorporated for object tracking. Our algorithm localizes the tracked object in each video frame using a rectangular window. A state $s$ is a length-5 vector, $s = (x, y, \theta, w, h)$, that parameterizes the windows position $(x, y)$, orientation $(\theta)$ and width and height $(w, h)$. The proposed algorithm is based on maximum likelihood estimate (i.e., the most probable location of the object) given all the observations up to that time instance, $s_t^* = \arg\max_{s_t} p(s_t|O_t)$.

We assume that state transition is a Gaussian distribution, i.e.,

$$p(s_t|s_{t-1}) \sim \mathcal{N}(s_{t-1}, \Sigma_s) \tag{16}$$

where $\Sigma_s$ is a diagonal matrix. According to this distribution, the tracker then draws $N$ samples $S_t = \{c_1, \ldots, c_N\}$ which represent the possible locations of the target. Denote $y_t^i$ as the appearance vector of $o_t$, and $Y_t = \{y_t^1, \ldots, y_t^N\}$ as a set of appearance vectors that corresponds to the set of state vectors $S_t$. The posterior probability that the tracked object is at $c_i$ in video frame $o_t$ is then defined as

$$p(s_t = c_i|O_t) = \kappa p(y_t^i|V, W, \mu, \sigma)p(s_t = c_i|s_{t-1}^*) \tag{17}$$

where $\kappa$ is a constant. Therefore, $s_t^* = \arg\max_{c_i \in S_t} p(s_t = c_i|O_t)$.

Once $s_t^*$ is determined, the corresponding observation $y_t^*$ will be a new example to update $W$ and $\mu$. Appearance vectors $y_t^i$ with large $p(y_t^i|V, W, \mu, \sigma)$ but whose corresponding state parameters $c_i$ are away from $s_t^*$ will be used as new examples to update $V$.

Our tracking assumes $o_1$ and $s_1^*$ are given (through object detection) and thus obtains the first appearance vector $y_1$ which in turn is used an the initial value of $\mu$, but $V$ and $W$ are

unknown at the outset. When $V$ and $W$ are not available, our tracking algorithm is based on template matching (with $\mu$ being the template). The matrix $W$ is computed after a small number of appearance vectors are observed. When $W$ is available, we can then start to compute and update $V$ accordingly.

As mentioned in the Section 3.1.1, it is difficult to obtain an accurate estimate of $\sigma$. In our tracking the system, we adaptively adjust $\sigma$ according to $\Sigma_m$ in $W$. We set $\sigma$ be a fixed fraction of the smallest eigenvalues in $\Sigma_m$. This will ensure the distance measurement in (6) will not be biased to favor either the Mahalanobis distance in the subspace or the distance to the subspace.

## 5   Experimental Results

We tested the proposed algorithm with numerous object tracking experiments. To examine whether our model is able to adapt and track objects in the dynamically changing environment, we recorded videos containing appearance deformation, large illumination change, and large pose variations. All the image sequences consist of $320 \times 240$ pixel grayscale videos, recorded at 30 frames/second and 256 gray-levels per pixel. The forgetting term is empirically selected as 0.85, and the batch size for update is set to 5 as a trade-off of computational efficiency as well as effectiveness of modeling appearance change due to fast motion. More experimental results and videos can be found at `http://www.ifp.uiuc.edu/~rlin1/adgm.html`.

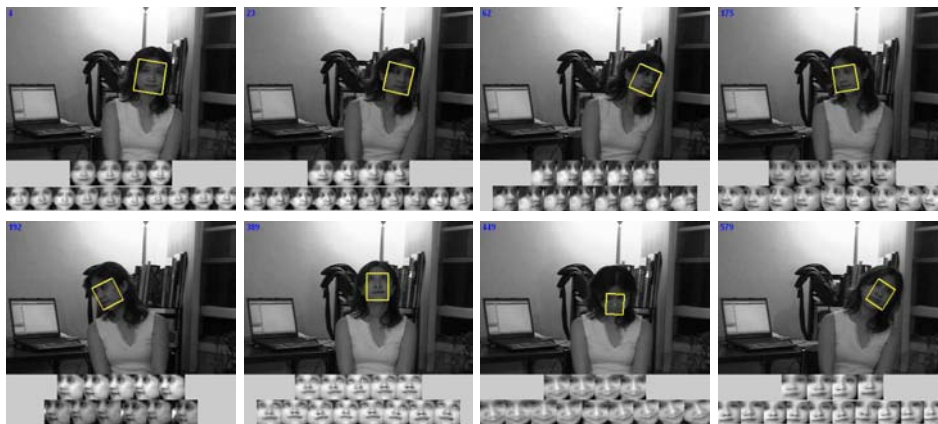

Figure 1: A target undergoes pose and lighting variation.

Figures 1 and 2 show snapshots of some tracking results enclosed with rectangular windows. There are two rows of images below each video frame. The first row shows the sampled images in the current frame that have the largest likelihoods of being the target locations according our discriminative generative model. The second row shows the sample images in the current video frame that are selected online for updating the discriminative generative model.

The results in Figure 1 show the our method is able to track targets undergoing pose and lighting change. Figure 2 shows tracking results where the object appearances change significantly due to variation in pose and lighting as well as cast shadows. These experiments demonstrate that our tracking algorithm is able to follow objects even when there is a large appearance change due to pose or lighting variation. We have also tested these two sequences with conventional view-based eigentracker [2] or template-based method. Empirical results show that such methods do not perform well as they do not update the object representation to account for appearance change.

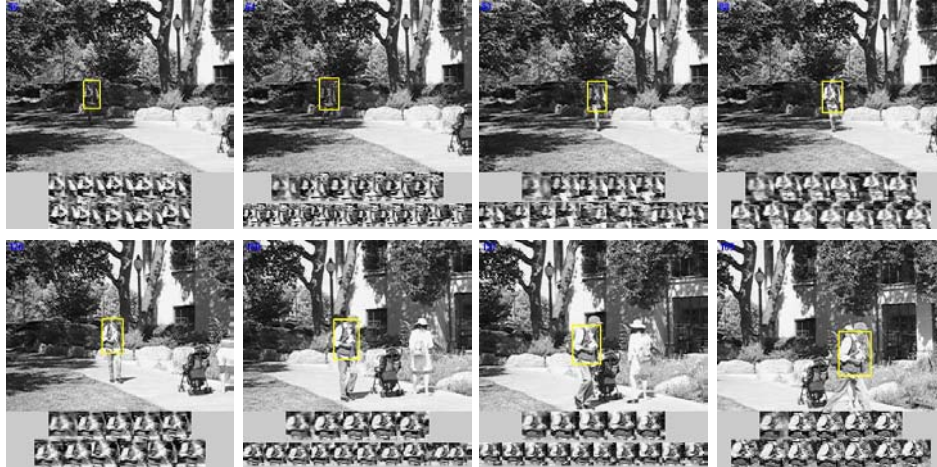

Figure 2: A target undergoes large lighting and pose variation with cast shadows.

## 6  Conclusion

We have presented a discriminative generative framework that generalizes the conventional Fisher Linear Discriminant algorithm with a proper probabilistic interpretation. For object tracking, we aim to find a discriminative generative model that best separates the target class from the background. With a computationally efficient algorithm that constantly update this discriminative model as time progresses, our method adapts the discriminative generative model to account for appearance variation of the target and background, thereby facilitating the tracking task in different situations. Our experiments show that the proposed model is able to learn a discriminative generative model for tracking target objects undergoing large pose and lighting changes. We also plan to apply the proposed method to other problems that deal with non-stationary data stream in our future work.

## References

[1] T. W. Anderson. *An Introduction to Multivariate Statistical Analysis*. Wiley, New York, 1984.

[2] M. J. Black and A. D. Jepson. Eigentracking: Robust matching and tracking of articulated objects using view-based representation. In B. Buxton and R. Cipolla, editors, *Proceedings of the Fourth European Conference on Computer Vision*, LNCS 1064, pp. 329–342. Springer Verlag, 1996.

[3] R. T. Collins and Y. Liu. On-line selection of discriminative tracking features. In *Proceedings of the Ninth IEEE International Conference on Computer Vision*, volume 1, pp. 346–352, 2003.

[4] G. H. Golub and C. F. Van Loan. *Matrix Computations*. The Johns Hopkins University Press, 1996.

[5] M. Isard and A. Blake. Contour tracking by stochastic propagation of conditional density. In B. Buxton and R. Cipolla, editors, *Proceedings of the Fourth European Conference on Computer Vision*, LNCS 1064, pp. 343–356. Springer Verlag, 1996.

[6] M. I. Jordan, editor. *Learning in Graphical Models*. MIT Press, 1999.

[7] A. Levy and M. Lindenbaum. Sequential Karhunen-Loeve basis extraction and its application to images. *IEEE Transactions on Image Processing*, 9(8):1371–1374, 2000.

[8] R.-S. Lin, D. Ross, J. Lim, and M.-H. Yang. Incremental subspace update with running mean. Technical report, Beckman Institute, University of Illinois at Urbana-Champaign, 2004. available at http://www.ifp.uiuc.edu/~rlinl/isuwrm.pdf.

[9] D. Ross, J. Lim, and M.-H. Yang. Adaptive probabilistic visual tracking with incremental subspace update. In T. Pajdla and J. Matas, editors, *Proceedings of the Eighth European Conference on Computer Vision*, LNCS 3022, pp. 470–482. Springer Verlag, 2004.

[10] M. E. Tipping and C. M. Bishop. Probabilistic principal component analysis. *Journal of the Royal Statistical Society, Series B*, 61(3):611–622, 1999.
